# Nonparametric sparse hierarchical models describe V1 fMRI responses to natural images

**Pradeep Ravikumar, Vincent Q. Vu and Bin Yu**
Department of Statistics
University of California, Berkeley
Berkeley, CA 94720-3860

**Thomas Naselaris, Kendrick N. Kay and Jack L. Gallant**
Department of Psychology
University of California, Berkeley
Berkeley, CA

## Abstract

We propose a novel hierarchical, nonlinear model that predicts brain activity in area V1 evoked by natural images. In the study reported here brain activity was measured by means of functional magnetic resonance imaging (fMRI), a noninvasive technique that provides an indirect measure of neural activity pooled over a small volume ($\approx$ 2mm cube) of brain tissue. Our model, which we call the V-SPAM model, is based on the reasonable assumption that fMRI measurements reflect the (possibly nonlinearly) pooled, rectified output of a large population of simple and complex cells in V1. It has a hierarchical filtering stage that consists of three layers: model simple cells, model complex cells, and a third layer in which the complex cells are linearly pooled (called "pooled-complex" cells). The pooling stage then obtains the measured fMRI signals as a *sparse additive model* (SpAM) in which a sparse nonparametric (nonlinear) combination of model complex cell and model pooled-complex cell outputs are summed. Our results show that the V-SPAM model predicts fMRI responses evoked by natural images better than a benchmark model that only provides linear pooling of model complex cells. Furthermore, the spatial receptive fields, frequency tuning and orientation tuning curves of the V-SPAM model estimated for each voxel appears to be consistent with the known properties of V1, and with previous analyses of this data set. A visualization procedure applied to the V-SPAM model shows that most of the nonlinear pooling consists of simple compressive or saturating nonlinearities.

## 1 Introduction

An important step toward understanding the neural basis of vision is to develop computational models that describe how complex visual stimuli are mapping onto evoked neuronal responses. This task is made challenging in part by the inherent difficulty of obtaining neurophysiological recordings from single neurons *in vivo*. An alternative approach is to base models on brain activity measured by means of functional magnetic resonance imaging (fMRI). fMRI measures changes in blood oxygenation and flow throughout the brain that occur as a consequence of metabolic demands. Although the relationship between measured fMRI activity and the spiking activity of neurons is rather complex, as a first-order approximation the fMRI signal can be considered to be monotonically related to the pooled activity of the underlying neural population.

In this paper we consider the task of predicting fMRI brain activity evoked by a series of gray-scale natural images. Natural images are a useful stimulus set for efficiently probing the visual system, because they are likely to evoke response from both early visual areas and from more central, highly nonlinear visual areas. The fMRI scanner provides a three-dimensional image of the brain with a spatial resolution of a few cubic millimeters and fairly low temporal resolution (about 0.5–1 Hz). After pre-processing the fMRI signals are represented as a vector of three-dimensional volume elements called voxels. Here we restrict our analysis to voxels sampled from visual area V1, the primary visual area in humans.

There are two problems that make predicting evoked responses of fMRI voxels difficult. First, fMRI signals are noisy and non-stationary in time. Second, each voxel reflects the combined influence of hundreds of thousands of neurons [4]. fMRI scans of a single voxel in human V1 likely reflect the nonlinearly-pooled, rectified outputs of two functionally distinct classes of neurons: simple cells that are sensitive to spatial phase, and phase-invariant complex cells [2]. Even if an accurate predictive model is obtained, there remains the issue of interpretability. It is not sufficient to construct a model that provides good predictions but whose function remains opaque (*i.e.*, a black box). In order for a predictive model to advance our understanding of the brain, the function of any predictive model must be conceptually interpretable.

In this paper we propose a new model that aims to overcome some of these problems. Our *V-SPAM model* is a hierarchical and sparse nonparametric additive model. It combines a biologically-inspired hierarchical filtering scheme with a nonlinear (nonparametric) pooling of the outputs from various levels of the hierarchical filtering stage. The model is estimated separately for each recorded fMRI voxel using a fit data set, and then its predictions are evaluated against an entirely separate data set reserved for this purpose.

The filtering component of the model consists of three distinct layers: simple cells, complex cells, and linear combinations of the complex cells (here called *pooled-complex cells*). The fMRI response is then modeled as a sparse additive combination of nonlinear (nonparametric) functions of the complex and pooled-complex cell model outputs. This last step automatically learns the optimal combinatorial output nonlinearity of the hierarchical filtering stage, and so permits us to model nonlinear V1 responses not captured by the simple and complex cell model components alone [6].

The fMRI dataset used in this paper was collected as part of an earlier study by [5]. That study also used a filtering model to describe the relationship between natural images and evoked fMRI signals, and used the estimated models in turn to decode (identify) images. However, the earlier study only provided linear pooling of model complex cell filters. Our results show that the V-SPAM model predicts fMRI responses evoked by natural images better than does the earlier linear pooling model. Furthermore, the spatial receptive fields, frequency tuning and orientation tuning curves of the V-SPAM model estimated for each voxel appear to be consistent with the known properties of V1, and with the previous results [5].

## 2 Background

### 2.1 Sparse Additive Models

The regression task consists of estimating the regression function $\mathbb{E}(Y|X)$ for a real-valued response $Y \in \mathbb{R}$ and a predictor-vector $X = (X_1, \ldots, X_p) \in \mathbb{R}^p$ from data $\{(X_i, Y_i),\ i = 1, \ldots n\}$.

In the nonparametric regression model, the response $Y_i = m(X_i) + \epsilon_i$, where $m$ is a general smooth function. Estimating this function (*i.e.*, smoothing) becomes challenging when the number of predictors $p$ is large. Even estimating linear models of the form $Y_i = \beta^\top X_i + \epsilon_i$, is challenging in these high-dimensional settings. For linear models however, when the vector $\beta$ is sparse, Tibshirani [8] and others have shown that the $\ell_1$ penalized estimator (also called the Lasso), $\hat{\beta} = \arg\min_\beta \sum_i (Y_i - \beta^\top X_i)^2 + \lambda \sum_{j=1}^p |\beta_j|$ can estimate a sparse model and has strong theoretical properties.

The sparse additive model (SpAM) framework of Ravikumar et al [7] extends these sparse linear models to the nonparametric domain. In additive models, introduced by Hastie and Tibshirani [3], the response $Y$ is an additive combination of *functions* of the predictors, $Y = \sum_{j=1}^p f_j(X_j) + \epsilon$ Here the functions $\{f_j\}$ are constrained to lie in a class of smooth functions, such as the space of

functions with square integrable double derivatives (*i.e.*, the Sobolev space of order two). A sparse additive model then imposes a sparsity constraint on the set $J = \{j : f_j \not\equiv 0\}$ of functions $f_j$ that are nonzero.

## 2.2 Fitting Algorithm for Sparse Additive Models

The paper [7] proposes a fitting procedure for sparse additive models that has good statistical properties even in the large $p$ small $n$ regime. Their SpAM fitting algorithm is summarized in Figure 1. It performs a coordinate descent (in the $L_2(P^n)$ space, with $P^n$ the sample distribution). At each step the algorithm performs nonparametric regression of the current residual onto a single predictor, and then does a soft threshold.

---

*Input*: Data $(X_i, Y_i)$, regularization parameter $\lambda$.

*Initialize* $f_j = f_j^{(0)}$, for $j = 1, \ldots, p$.

*Iterate* until convergence:

    *For each* $j = 1, \ldots, p$:

        Compute the residual: $\mathcal{R}_j = Y - \sum_{k \neq k} f_k(X_k)$;

        Estimate the conditional expectation $\mathcal{P}_j = \mathbb{E}[\mathcal{R}_j | X_j]$ by smoothing: $\hat{\mathcal{P}}_j = \mathcal{S}_j \mathcal{R}_j$;

        Set $s_j^2 = n^{-1} \sum_{i=1}^n \hat{\mathcal{P}}_j^2(i)$.

        Soft-threshold: $f_j = [1 - \lambda/\hat{s}_j]_+ \hat{\mathcal{P}}_j$;

        Center: $f_j \leftarrow f_j - \text{mean}(f_j)$.

*Output*: Component functions $f_j$ and estimator $\hat{m}(X_i) = \sum_j f_j(X_{ij})$.

---

Figure 1: THE SPAM BACKFITTING ALGORITHM

## 3 A model for pooled neural activity of voxels

Our V-SPAM model combines a biologically-inspired filtering scheme and a novel algorithm that permits nonlinear pooling of the outputs of the filtering stage. The filtering stage itself consists of three distinct layers, arranged hierarchically: simple cells, complex cells, and linear combinations of the complex cells (here called pooled-complex cells). The output of this filtering operation is then fed to an algorithm that estimates a nonlinear pooling function that optimizes predictive power.

### 3.1 Simple Cell Model

The first stage of the hierarchical filter is inspired by simple cells that are known to exist in area V1. The receptive fields of V1 simple cells are known to be generally consistent with a Gabor wavelet model [6]. Most importantly, they are spatially localized, oriented, spatial frequency band-pass and phase selective. (see Figure 2.)

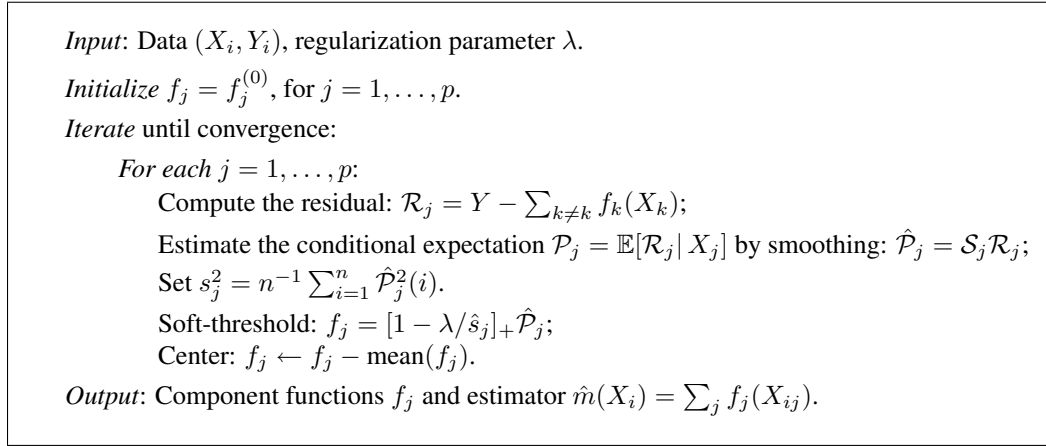

Figure 2: Gabor wavelets. Each row shows a family of Gabor wavelets that share a common spatial location and frequency, but differ in orientation. This is only a small fraction of all of the wavelets in the pyramid.

In our model the simple cell filter bank was implemented as a Gabor wavelet pyramid, as follows. Let $I$ denote an image, and $d$ the number of pixels. It can thus be represented as a pixel vector in $\mathbb{R}^d$. Denote by $\phi_j$ a Gabor wavelet sampled on a grid the size of the image, so that it too can be represented as vector in $\mathbb{R}^d$. Then our simple cell model, for the activation given the image $I$ as stimulus, is given by, $X_j(I) = [\langle \phi_j, I \rangle]_+$, where $\langle \cdot \rangle$ is the Euclidean inner product, and $[\ ]_+$ is a non-negative rectification. (See Figure 3.) Correspondingly, $X_j(I) = [\langle \psi_j, I \rangle]_+$ gives the activation of the $180^\circ$ spatial phase counterpart.

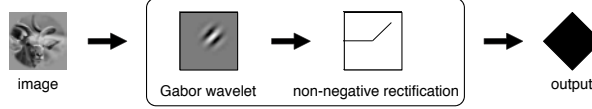

Figure 3: Simple cell model. The activation of a model simple cell given an image is the inner product of the image with a Gabor wavelet, followed by a non-negative rectification.

## 3.2 Complex Cell Model

The second stage of the hierarchical filter is inspired by complex cells that are also known to exist in area V1. Complex cells are similar to simple cells, except they are not sensitive to spatial phase. In our model the complex cell filter bank was implemented by taking the sum of squares of the outputs of four simple cells (corresponding to the wavelet pairs that are identical up to phase), followed by a fixed output nonlinearity. The activation of the model complex cell given an image $I$ is given by,

$$X_j(I) = \log(1 + \overline{[\langle \phi_j, I \rangle]_+^2 + [\langle \psi_j, I \rangle]_+^2 + [\langle \phi_j, I \rangle]_+^2 + [\langle \psi_j, I \rangle]_+^2}) \qquad (1)$$

$$= \log(1 + \overline{[\langle \phi_j, I \rangle]^2 + [\langle \psi_j, I \rangle]^2}) \qquad (2)$$

where $\phi_j$ and $\psi_j$ are Gabor wavelets identical up to phase (also called a quadrature pair; see Figure 4).

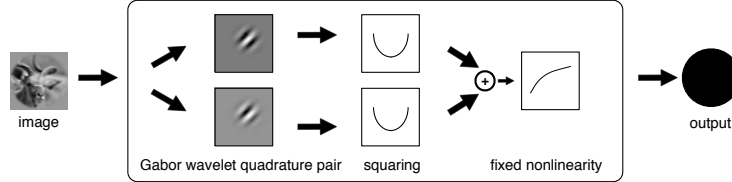

Figure 4: Complex cell model. The activation of a model complex cell given an image is the sum of squares of the inner products of the image with a quadrature pair of Gabor wavelets followed by a nonlinearity. This is equivalently modeled by summing the squares of 4 simple cell model outputs, followed by a nonlinearity.

## 3.3 Pooled-complex Cell Model

The hierarchical filtering component of our model also includes a third filtering stage, linear pooling of complex cells sharing a common spatial location and frequency. This stage has no direct biological interpretation in terms of area V1, but has been included to improve representational power of the model: a linear combination of complex cells (the pooled-complex cell), followed by a nonlinearity, cannot be expressed as an additive combination of nonlinear functions of individual complex cells. Note that this element might be particularly useful for modeling responses in higher visual areas beyond V1.

If $X_{j_1} \ldots X_{j_k}$ correspond to complex cells with the same spatial location and frequency, then the corresponding pooled-complex cell (which thus sums over different orientations) is given by, $Z_{j_1 \ldots j_k} = \sum_{l=1}^{k} X_{j_l}$. (See Figure 5.)

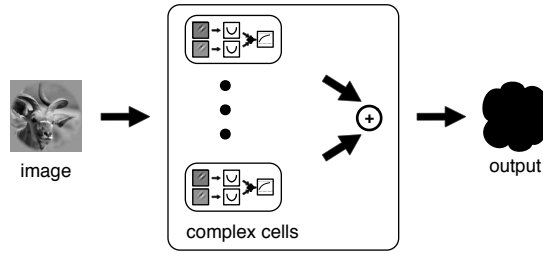

image → complex cells → output

Figure 5: Pooled-complex cell model. Subsets of complex cells that share a common spatial location and frequency are summed.

## 3.4  V-SPAM model

Finally, the predicted fMRI response $Y$ is obtained as a sparse additive combination of complex cell and pooled-complex cell outputs. Denote the complex cell outputs by $X_1 \ \cdots \ X_p$, and the pooled-complex cell outputs by $Z_1 \ \cdots \ Z_q$. Then the fMRI response $Y$ is modeled as a sparse additive (nonparametric) model, $Y = \sum_{j=1}^{p} f_j(X_j) + \sum_{l=1}^{q} g_l(Z_l) + \phi$ Figure 6 summarizes the entire V-SPAM model, including both filtering and pooling components.

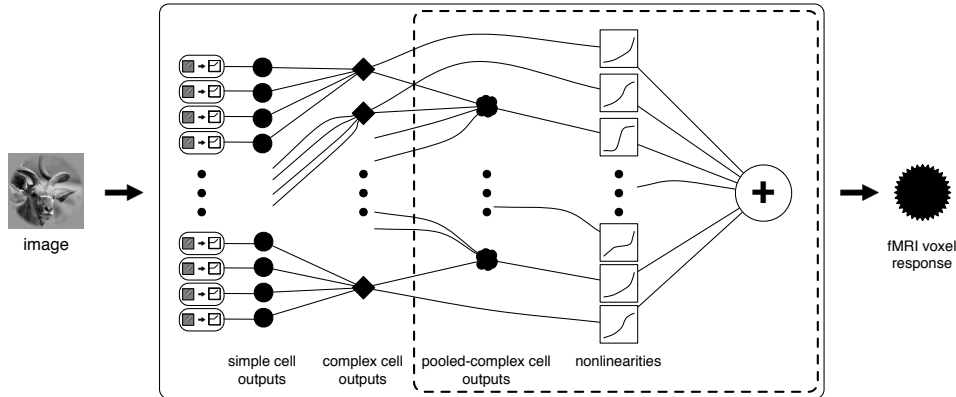

image → simple cell outputs → complex cell outputs → pooled-complex cell outputs → nonlinearities → + → fMRI voxel response

Figure 6: V-SPAM model. The fMRI voxel response is modeled as the summation of nonlinear functions of complex and pooled-complex cell outputs. The connections and components in the dashed region are to be estimated from the data under the assumption that many of them are null.

## 4  Experiments

### 4.1  Data description

The data set analyzed in this paper consists of a total of 1,294 voxels recorded from area V1 of one human observer. A 4T Varian MRI scanner provided voxels of size 2mm x 2mm x 2.5mm at a frequency of 1Hz. The visual stimuli used in the experiment consisted of 1,750 20-by-20 degree grayscale natural images, masked by a circular aperture. A two-stage procedure was used for data collection. In the first stage, 1,750 natural images were presented to the subject 2 times each. This data set was used to fit the model. In the second stage, 120 additional natural images were presented 13 times each. This data set was used for model validation. (Note that the images used for estimation and validation were distinct.) In all cases images were flashed briefly 3 times during a 1 second display period, and there was a blank period of 3 seconds between successive images. After acquisition the fMRI signals were pre-processed to reduce temporal non-stationarity and increase signal-to-noise [5]. Complete details of the fMRI experiment can be found in [5].

## 4.2 V-SPAM model fitting

The V-SPAM model was fitted separately for each of the 1,294 voxels using the training set of 1,750 images and the evoked fMRI responses. The fitting procedure can be conceptualized in four successive stages that roughly parallel the hierarchical layers of the model itself.

In the first stage, the model complex cell outputs are computed according to equation (2) using a pyramid (or family) of Gabor wavelets sampled on a grid of 128 x 128 pixels. The pyramid includes 5 spatial frequencies (or scales): 1, 2, 4, 8, 16, and 32 cycles/field of view. At each spatial frequency $\omega$ the wavelets are positioned evenly on a $\omega \times \omega$ grid covering the image. All combinations of 8 orientations and 2 phases occur at each of the $\omega \times \omega$ positions. In total, the pyramid consists of 10,920 quadrature pairs plus 1 constant wavelet (corresponding to mean luminance).

In the second stage, the model complex cell outputs are pre-screened in order to eliminate complex cell outputs that are unrelated to a voxel's response, and to reduce the computational complexity of successive stages of fitting. This is accomplished by considering the squared-correlation of the response of each complex cell with the evoked voxel response, using the 1,750 images in the training set. Only the top $k$ complex cells are retained. In pilot studies we found empirically that $k = 100$ was enough to give good statistical and computational performance (data not shown).

In the third stage, pooled-complex cells (see Section 3) are formed from the complex cell outputs that passed the pre-screening in fitting stage 2.

In the fourth and final stage, the complex and pooled-complex cell responses to the images in the training set are used as predictors in the SpAM fitting algorithm (see Figure 1), and this is optimized to fit the voxel responses evoked by the same 1,750 images in the training set. The smoothing is done by means of Gaussian kernel regression with plug-in bandwidth, and the regularization parameter is selected by the Akaike information criterion (AIC).

## 4.3 Model validation

For each voxel, we evaluate the fitted V-SPAM models by computing the *predictive* $R^2$ (squared correlation) of the predicted and actual fMRI responses evoked by each of the 120 images in the validation set.

To permit a more complete evaluation of the V-SPAM model, we used the same data to fit a simpler model more directly comparable to the one used in earlier work with this data set [5]. The sparse linear pooling model aims to predict each voxel's response as a linear combination of all 10,921 estimated complex cell outputs. This model has the form, $Y(I) = \beta_0 + \sum_{j=1}^{p} \beta_j X_j(I) + \epsilon$, where the $X_j(I)$ are the complex cell outputs estimated according to (2), with the $p = 10,921$ Gabor wavelets described in Section 4.2. The coefficients $\beta_j$, $j = 0, \ldots, p$, were estimated by L2 Boosting [1] with the stopping criterion determined by 5-fold cross-validation within the same data set. This model is a sparsified version of the one used in [5], and has comparable prediction performance.

## 5 Results

Figure 7 (left) shows a scatterplot comparing the performance of the V-SPAM model with that of the sparse linear pooling model for all 1,294 voxels. The vertical axis gives performance of the V-SPAM model, and the horizontal axis the sparse linear pooling model. Each point corresponds to a single voxel. The inset region contains 429 voxels for which both models had some predictive power ($R^2 \geq 0.1$). For these voxels, the relative improvement of the V-SPAM model over the sparse linear pooling model is shown in the histogram to the right. The predictions of the V-SPAM model were on average 14% better than those of the sparse linear pooling model (standard deviation 17%).

### 5.1 Estimated receptive fields and tuning curves

Figure 8 shows the spatial receptive-fields (RF's) and joint frequency and orientation tuning curves estimated using the V-SPAM model for 3 voxels. These voxels were chosen because they had high predictive power ($R^2$'s of 0.65, 0.59, and 0.63, respectively from left to right) and so were modeled accurately. The upper row of the figure shows the spatial RF of each voxel. The intensity at each

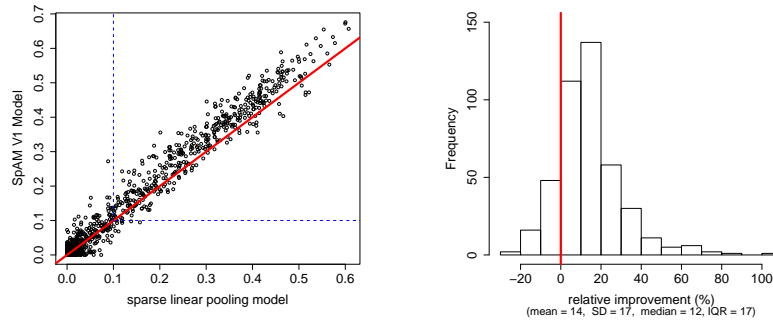

Figure 7: Predictive $R^2$ of the fitted V-SPAM model compared against the fitted sparse linear pooling model. (Left) Each of the 1,294 points in the scatterplot corresponds to a single voxel. (Right) Relative performance for the 429 voxels contained in the inset region on the left.

location in the spatial RF represents the standardized predicted response of the voxel to an image stimulus consisting of a single pixel at that location. The spatial RF's of these voxels are clearly localized in space, consistent with the known retinotopic organization of V1 and previous fMRI results [9]. The lower row of Figure 8 shows the joint frequency and orientation tuning properties of these same 3 voxels. Here the tuning curves were estimated by computing the predicted response of the fitted voxel model to cosine gratings of varying orientation (degrees) and spatial frequency (cycles/field of view). All of the voxels are tuned to spatial frequencies above about 8 cycles/field of view, while orientation tuning varies from voxel to voxel. The joint spatial frequency and orientation tuning of all 3 voxels appears to be non-separable (i.e. their orientation tuning is not a constant function of frequency).

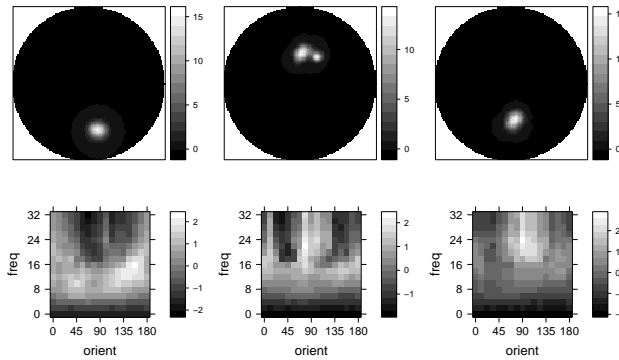

Figure 8: (upper) Spatial receptive-fields (RF's) and (lower) joint frequency and orientation tuning curves estimated by the V-SPAM model for 3 voxels with high predictive power ($R^2$'s of 0.65, 0.59, 0.63, left to right). Each location in the spatial RF shows the standardized predicted response of the voxel to an image consisting of a single pixel at that location. The tuning curves show the standardized predicted response of the voxel to cosine gratings of varying orientation (degrees) and spatial frequency (cycles/field of view).

## 5.2 Nonlinearities

One of the potential advantages of the V-SPAM model over other approaches is that it can reveal novel nonlinear tuning and pooling properties, as revealed by the nonlinear summation occurring in the final stage of the V-SPAM model. Figure 9 illustrates some of these functions estimated for a typical voxel with high predictive power ($R^2$ of 0.63). These correspond to the nonlinearities appearing in the final stage of the V-SPAM model (see Figure 6). Here the horizontal axis is the input in standard units of the corresponding model complex or pooled-complex cell outputs, and the vertical axis is the output in standard units of predicted responses. For this voxel, these are the

4 largest (ranked by $L^2$ norm) nonlinearities. All 4 of these nonlinearities are compressive. The remaining 75 nonlinearities present in the voxel's fitted model have similar shapes, but are much smaller and hence contribute less to the predicted response. They are overlaid in the final panel of Figure 9.

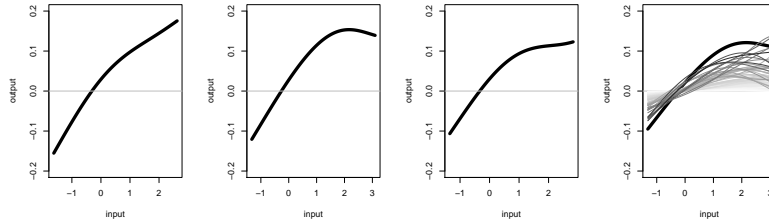

Figure 9: Nonlinearities estimated in the V-SPAM model for a voxel with high predictive power ($R^2$: 0.63). The 4 largest (ranked by $L^2$ norm) are shown left to right by the thick lines. The other 75 nonlinearities for this voxel (overlaid in the right panel) are smaller and contribute less to the predicted response.

## 6  Discussion and conclusions

Our V-SPAM model provides better predictions of fMRI activity evoked by natural images than does a sparse linear model similar to that used in an earlier study of this data set [5]. This increased predictive power of the V-SPAM model reflects the fact that it can describe explicitly the nonlinear pooling that likely occurs among the many neurons whose pooled activity contributes to measured fMRI signals. These pooled output nonlinearities are likely a critical component of nonlinear computation across the visual hierarchy. Therefore, the SpAM framework may be particularly useful for modeling neurons or fMRI signals recorded in higher and more nonlinear stages of visual processing beyond V1.

## References

[1] Peter Bühlmann and Bin Yu. Boosting with the l2 loss: Regression and classification. *Journal of the American Statistical Association*, 98(462):324–339, 2003.

[2] R.L. De Valois and K. K. De Valois. *Spatial Vision*. Oxford University Press, 1990.

[3] Trevor Hastie and Robert Tibshirani. *Generalized additive models*. Chapman & Hall Ltd., 1999.

[4] D. J. Heeger, A. C. Huk, W. S. Geisler, and D. G. Albrecht. Spikes versus bold: what does neuroimaging tell us about neuronal activity? *Nat Neurosci*, 3(7):631–633, 2000.

[5] Kendrick N. Kay, Thomas Naselaris, Ryan J. Prenger, and Jack L. Gallant. Identifying natural images from human brain activity. *Nature*, 452(7185):352–355, 2008.

[6] Bruno A. Olshausen and David J. Field. Emergence of simple-cell receptive field properties by learning a sparse code for natural images. *Nature*, 381(6583):607–609, June 1996.

[7] Pradeep Ravikumar, Han Liu, John Lafferty, and Larry Wasserman. Spam: Sparse additive models. *Neural Information Processing Systems*, 2007.

[8] R. Tibshirani. Regression shrinkage and selection via the lasso. *J. Royal. Statist. Soc B.*, 58, No. 1:267–288, 1996.

[9] Brian A. Wandell, Serge O. Dumoulin, and Alyssa A. Brewer. Visual field maps in human cortex. *Neuron*, 56(2):366–383, 2007.

